# A PAC-Bayes approach to the Set Covering Machine

**François Laviolette, Mario Marchand**
IFT-GLO, Université Laval
Sainte-Foy (QC) Canada, G1K-7P4
*given_name.surname@ift.ulaval.ca*

**Mohak Shah**
SITE, University of Ottawa
Ottawa, Ont. Canada,K1N-6N5
*mshah@site.uottawa.ca*

## Abstract

We design a new learning algorithm for the Set Covering Machine from a PAC-Bayes perspective and propose a PAC-Bayes risk bound which is minimized for classifiers achieving a non trivial margin-sparsity trade-off.

## 1   Introduction

Learning algorithms try to produce classifiers with small prediction error by trying to optimize some function that can be computed from a training set of examples and a classifier. We currently do not know exactly what function should be optimized but several forms have been proposed. At one end of the spectrum, we have the set covering machine (SCM), proposed by Marchand and Shawe-Taylor (2002), that tries to find the sparsest classifier making few training errors. At the other end, we have the support vector machine (SVM), proposed by Boser et al. (1992), that tries to find the maximum soft-margin separating hyperplane on the training data. Since both of these learning machines can produce classifiers having small prediction error, we have recently investigated (Laviolette et al., 2005) if better classifiers could be found by learning algorithms that try to optimize a non-trivial function that depends on both the sparsity of a classifier and the magnitude of its separating margin. Our main result was a general data-compression risk bound that applies to any algorithm producing classifiers represented by two complementary sources of information: a subset of the training set, called the *compression set*, and a *message string* of additional information. In addition, we proposed a new algorithm for the SCM where the information string was used to encode radius values for data-dependent balls and, consequently, the location of the decision surface of the classifier. Since a small message string is sufficient when large regions of equally good radius values exist for balls, the data compression risk bound applied to this version of the SCM exhibits, *indirectly*, a non-trivial margin-sparsity trade-off. Moreover, this version of the SCM currently suffers from the fact that the radius values, used in the final classifier, depends on a *a priori* chosen distance scale $R$. In this paper, we use a new PAC-Bayes approach, that applies to the sample-compression setting, and present a new learning algorithm for the SCM that does not suffer from this scaling problem. Moreover, we propose a risk bound that depends more explicitly on the margin and which is also minimized by classifiers achieving a non-trivial margin-sparsity trade-off.

## 2  Definitions

We consider binary classification problems where the input space $\mathcal{X}$ consists of an arbitrary subset of $\mathbb{R}^n$ and the output space $\mathcal{Y} = \{0, 1\}$. An example $\mathbf{z} \stackrel{\text{def}}{=} (\mathbf{x}, y)$ is an input-output pair where $\mathbf{x} \in \mathcal{X}$ and $y \in \mathcal{Y}$. In the probably approximately correct (PAC) setting, we assume that each example $\mathbf{z}$ is generated independently according to the same (but unknown) distribution $D$. The (true) *risk* $R(f)$ of a classifier $f : \mathcal{X} \to \mathcal{Y}$ is defined to be the probability that $f$ misclassifies $\mathbf{z}$ on a random draw according to $D$:

$$R(f) \stackrel{\text{def}}{=} \Pr_{(\mathbf{x}, y) \sim D}\left( f(\mathbf{x}) \neq y \right) = \mathbf{E}_{(\mathbf{x}, y) \sim D} I(f(\mathbf{x}) \neq y)$$

where $I(a) = 1$ if predicate $a$ is true and 0 otherwise. Given a training set $S = (\mathbf{z}_1, \ldots, \mathbf{z}_m)$ of $m$ examples, the task of a learning algorithm is to construct a classifier with the smallest possible risk without any information about $D$. To achieve this goal, the learner can compute the *empirical risk* $R_S(f)$ of any given classifier $f$ according to:

$$R_S(f) \stackrel{\text{def}}{=} \frac{1}{m} \sum_{i=1}^{m} I(f(\mathbf{x}_i) \neq y_i) \stackrel{\text{def}}{=} \mathbf{E}_{(\mathbf{x}, y) \sim S} I(f(\mathbf{x}) \neq y)$$

We focus on learning algorithms that construct a *conjunction (or disjunction)* of features called *data-dependent balls* from a training set. Each *data-dependent ball* is defined by a *center* and a *radius* value. The center is an input example $\mathbf{x}_i$ chosen among the training set $S$. For any test example $\mathbf{x}$, the output of a ball $h$, of radius $\rho$ and centered on example $\mathbf{x}_i$, and is given by

$$h_{i,\rho}(\mathbf{x}) \stackrel{\text{def}}{=} \begin{cases} y_i & \text{if } d(\mathbf{x}, \mathbf{x}_i) \leq \rho \\ \bar{y}_i & \text{otherwise} \end{cases},$$

where $\bar{y}_i$ denotes the boolean complement of $y_i$ and $d(\mathbf{x}, \mathbf{x}_i)$ denotes the distance between the two points. Note that any metric can be used for the distance here.

To specify a *conjunction of balls* we first need to list all the examples that participate as centers for the balls in the conjunction. For this purpose, we use a vector $\mathbf{i} \stackrel{\text{def}}{=} (i_1, \ldots, i_{|\mathbf{i}|})$ of indices $i_j \in \{1, \ldots, m\}$ such that $i_1 < i_2 < \ldots < i_{|\mathbf{i}|}$ where $|\mathbf{i}|$ is the number of indices present in $\mathbf{i}$ (and thus the number of balls in the conjunction).

To complete the specification of a conjunction of balls, we need a vector $\boldsymbol{\rho} = (\rho_{i_1}, \rho_{i_2}, \ldots, \rho_{i_{|\mathbf{i}|}})$ of radius values where $i_j \in \{1, \ldots, m\}$ for $j \in \{1, \ldots, |\mathbf{i}|\}$.

On any input example $\mathbf{x}$, the output $C_{\mathbf{i}, \boldsymbol{\rho}}(\mathbf{x})$ of a conjunction of balls is given by:

$$C_{\mathbf{i}, \boldsymbol{\rho}}(\mathbf{x}) \stackrel{\text{def}}{=} \begin{cases} 1 & \text{if } h_{j, \rho_j}(\mathbf{x}) = 1 \quad \forall j \in \mathbf{i} \\ 0 & \text{if } \exists j \in \mathbf{i} : h_{j, \rho_j}(\mathbf{x}) = 0 \end{cases}$$

Finally, any algorithm that builds a conjunction can be used to build a disjunction just by exchanging the role of the positive and negative labelled examples. Due to lack of space, we describe here only the case of a conjunction.

## 3  A PAC-Bayes Risk Bound

The PAC-Bayes approach, initiated by McAllester (1999a), aims at providing PAC guarantees to "Bayesian" learning algorithms. These algorithms are specified in terms of a *prior distribution* $P$ over a space of classifiers that characterizes our

prior belief about good classifiers (before the observation of the data) and a *posterior distribution* $Q$ (over the same space of classifiers) that takes into account the additional information provided by the training data. A remarkable result that came out from this line of research, known as the "PAC-Bayes theorem", provides a tight upper bound on the risk of a stochastic classifier called the *Gibbs classifier*. Given an input example $\mathbf{x}$, the label $G_Q(\mathbf{x})$ assigned to $\mathbf{x}$ by the Gibbs classifier is defined by the following process. We first choose a classifier $h$ according to the posterior distribution $Q$ and then use $h$ to assign the label $h(\mathbf{x})$ to $\mathbf{x}$. The PAC-Bayes theorem was first proposed by McAllester (1999b) and later improved by others (see Langford (2005) for a survey). However, for all these versions of the PAC-Bayes theorem, the prior $P$ must be defined without reference to the training data. Consequently, these theorems cannot be applied to the sample-compression setting where classifiers are partly described by a subset of the training data (as for the case of the SCM).

In the sample compression setting, each classifier is described by a subset $S_{\mathbf{i}}$ of the training data, called the *compression set*, and a *message string* $\sigma$ that represents the additional information needed to obtain a classifier. In other words, in this setting, there exists a reconstruction function $\mathcal{R}$ that outputs a classifier $\mathcal{R}(\sigma, S_{\mathbf{i}})$ when given an arbitrary compression set $S_{\mathbf{i}}$ and a message string $\sigma$.

Given a training set $S$, the compression set $S_{\mathbf{i}} \subseteq S$ is defined by a vector of indices $\mathbf{i} \stackrel{\text{def}}{=} (i_1, \ldots, i_{|\mathbf{i}|})$ that points to individual examples in $S$. For the case of a conjunction of balls, each $j \in \mathbf{i}$ will point to a training example that is used for a ball center and the message string $\sigma$ will be the vector $\boldsymbol{\rho}$ of radius values (defined above) that are used for the balls. Hence, given $S_{\mathbf{i}}$ and $\boldsymbol{\rho}$, the classifier obtained from $\mathcal{R}(\boldsymbol{\rho}, S_{\mathbf{i}})$ is just the conjunction $C_{\mathbf{i}, \boldsymbol{\rho}}$ defined previously.[1]

Recently, Laviolette and Marchand (2005) have extended the PAC-Bayes theorem to the sample-compression setting. Their proposed risk bound depends on a data-independent prior $P$ and a data-dependent posterior $Q$ that are both defined on $\mathcal{I} \times \mathcal{M}$ where $\mathcal{I}$ denotes the set of the $2^m$ possible index vectors $\mathbf{i}$ and $\mathcal{M}$ denotes, in our case, the set of possible radius vectors $\boldsymbol{\rho}$. The posterior $Q$ is used by a stochastic classifier, called the *sample-compressed Gibbs classifier* $G_Q$, defined as follows. Given a training set $S$ and given a new (testing) input example $\mathbf{x}$, a sample-compressed Gibbs classifier $G_Q$ chooses randomly $(\mathbf{i}, \boldsymbol{\rho})$ according to $Q$ to obtain classifier $\mathcal{R}(\boldsymbol{\rho}, S_{\mathbf{i}})$ which is then used to determine the class label of $\mathbf{x}$.

In this paper we focus on the case where, given any training set $S$, the learner returns a Gibbs classifier defined with a posterior distribution $Q$ having all its weight on a single vector $\mathbf{i}$. Hence, a single compression set $S_{\mathbf{i}}$ will be used for the final classifier. However, the radius $\rho_i$ for each $i \in \mathbf{i}$ will be chosen stochastically according to the posterior $Q$. Hence we consider posteriors $Q$ such that $Q(\mathbf{i}', \boldsymbol{\rho}) = I(\mathbf{i} = \mathbf{i}')Q_{\mathbf{i}}(\boldsymbol{\rho})$ where $\mathbf{i}$ is the vector of indices chosen by the learner. Hence, given a training set $S$, the true risk $R(G_{Q_{\mathbf{i}}})$ of $G_{Q_{\mathbf{i}}}$ and its empirical risk $R_S(G_{Q_{\mathbf{i}}})$ are defined by

$$R(G_{Q_{\mathbf{i}}}) \stackrel{\text{def}}{=} \mathop{\mathbf{E}}_{\boldsymbol{\rho} \sim Q_{\mathbf{i}}} R(\mathcal{R}(\boldsymbol{\rho}, S_{\mathbf{i}})) \quad ; \quad R_S(G_{Q_{\mathbf{i}}}) \stackrel{\text{def}}{=} \mathop{\mathbf{E}}_{\boldsymbol{\rho} \sim Q_{\mathbf{i}}} R_{S_{\bar{\mathbf{i}}}}(\mathcal{R}(\boldsymbol{\rho}, S_{\mathbf{i}})) ,$$

where $\bar{\mathbf{i}}$ denotes the set of indices not present in $\mathbf{i}$. Thus, $\bar{\mathbf{i}} \cap \mathbf{i} = \emptyset$ and $\mathbf{i} \cup \bar{\mathbf{i}} = (1, \ldots, m)$.

In contrast with the posterior $Q$, the prior $P$ assigns a non zero weight to several vectors $\mathbf{i}$. Let $P_{\mathcal{I}}(\mathbf{i})$ denote the prior probability $P$ assigned to vector $\mathbf{i}$ and let $P_{\mathbf{i}}(\boldsymbol{\rho})$

denote the probability density function associated with prior $P$ given $\mathbf{i}$. The risk bound depends on the Kullback-Leibler divergence $\mathrm{KL}(Q\|P)$ between the posterior $Q$ and the prior $P$ which, in our case, gives

$$\mathrm{KL}(Q_{\mathbf{i}}\|P) = \mathop{\mathbf{E}}_{\boldsymbol{\rho} \sim Q_{\mathbf{i}}} \ln \frac{Q_{\mathbf{i}}(\boldsymbol{\rho})}{P_{\mathcal{I}}(\mathbf{i})P_{\mathbf{i}}(\boldsymbol{\rho})} \ .$$

For these classes of posteriors $Q$ and priors $P$, the PAC-Bayes theorem of Laviolette and Marchand (2005) reduces to the following simpler version.

**Theorem 1 (Laviolette and Marchand (2005))** *Given all our previous definitions, for any prior $P$ and for any $\delta \in (0,1]$*

$$\mathop{\mathrm{Pr}}_{S \sim D^m} \Big( \forall Q_{\mathbf{i}} : \ \mathrm{kl}(R_S(G_{Q_{\mathbf{i}}})\|R(G_{Q_{\mathbf{i}}})) \leq \ \tfrac{1}{m-|\mathbf{i}|} \ \big[\mathrm{KL}(Q_{\mathbf{i}}\|P) + \ln \tfrac{m+1}{\delta}\big] \Big) \geq 1 - \delta \ ,$$

*where*

$$\mathrm{kl}(q\|p) \overset{\mathrm{def}}{=} q \ln \frac{q}{p} + (1-q) \ln \frac{1-q}{1-p} \ .$$

To obtain a bound for $R(G_{Q_{\mathbf{i}}})$ we need to specify $Q_{\mathbf{i}}(\boldsymbol{\rho})$, $P_{\mathcal{I}}(\mathbf{i})$, and $P_{\mathbf{i}}(\boldsymbol{\rho})$.

Since all vectors $\mathbf{i}$ having the same size $|\mathbf{i}|$ are, *a priori*, equally "good", we choose

$$P_{\mathcal{I}}(\mathbf{i}) = \frac{1}{\binom{m}{|\mathbf{i}|}} p(|\mathbf{i}|)$$

for any $p(\cdot)$ such that $\sum_{d=0}^{m} p(d) = 1$. We could choose $p(d) = 1/(m+1)$ for $d \in \{0, 1, \ldots, m\}$ if we have complete ignorance about the size $|\mathbf{i}|$ of the final classifier. But since the risk bound will deteriorate for large $|\mathbf{i}|$, it is generally preferable to choose, for $p(d)$, a slowly decreasing function of $d$.

For the specification of $P_{\mathbf{i}}(\boldsymbol{\rho})$, we assume that each radius value, in some predefined interval $[0, R]$, is equally likely to be chosen for each $\rho_i$ such that $i \in \mathbf{i}$. Here $R$ is some "large" distance specified *a priori*. For $Q_{\mathbf{i}}(\boldsymbol{\rho})$, a margin interval $[a_i, b_i] \subseteq [0, R]$ of equally good radius values is chosen by the learner for each $i \in \mathbf{i}$. Hence, we choose

$$P_{\mathbf{i}}(\boldsymbol{\rho}) = \prod_{i \in \mathbf{i}} \frac{1}{R} \ = \ \left(\frac{1}{R}\right)^{|\mathbf{i}|} \quad ; \quad Q_{\mathbf{i}}(\boldsymbol{\rho}) = \prod_{i \in \mathbf{i}} \frac{1}{b_i - a_i} \ .$$

Therefore, the Gibbs classifier returned by the learner will draw each radius $\rho_i$ uniformly in $[a_i, b_i]$. A deterministic classifier is then specified by fixing each radius values $\rho_i \in [a_i, b_i]$. It is tempting at this point to choose $\rho_i = (a_i + b_i)/2 \ \forall i \in \mathbf{i}$ (*i.e.*, in the middle of each interval). However, we will see shortly that the PAC-Bayes theorem offers a better guarantee for another type of deterministic classifier.

Consequently, with these choices for $Q_{\mathbf{i}}(\boldsymbol{\rho})$, $P_{\mathcal{I}}(\mathbf{i})$, and $P_{\mathbf{i}}(\boldsymbol{\rho})$, the KL divergence between $Q_{\mathbf{i}}$ and $P$ is given by

$$KL(Q_{\mathbf{i}}\|P) \ = \ \ln \binom{m}{|\mathbf{i}|} + \ln \left(\frac{1}{p(|\mathbf{i}|)}\right) + \sum_{i \in \mathbf{i}} \ln \left(\frac{R}{b_i - a_i}\right) \ .$$

Notice that the KL divergence is small for small values of $|\mathbf{i}|$ (whenever $p(|\mathbf{i}|)$ is not too small) and for large margin values $(b_i - a_i)$. Hence, the KL divergence term in Theorem 1 favors both sparsity (small $|\mathbf{i}|$) and large margins. Hence, in practice, the minimum might occur for some $G_{Q_{\mathbf{i}}}$ that sacrifices sparsity whenever larger margins can be found.

Since the posterior $Q$ is identified by $\mathbf{i}$ and by the intervals $[a_i, b_i]$ $\forall i \in \mathbf{i}$, we will now refer to the Gibbs classifier $G_{Q_{\mathbf{i}}}$ by $G^{\mathbf{i}}_{\mathbf{ab}}$ where $\mathbf{a}$ and $\mathbf{b}$ are the vectors formed by the unions of $a_i$s and $b_i$s respectively. To obtain a risk bound for $G^{\mathbf{i}}_{\mathbf{ab}}$, we need to find a closed-form expression for $R_S(G^{\mathbf{i}}_{\mathbf{ab}})$. For this task, let $U[a, b]$ denote the uniform distribution over $[a, b]$ and let $\sigma^i_{a,b}(\mathbf{x})$ be the probability that a ball with center $\mathbf{x}_i$ assigns to $\mathbf{x}$ the class label $y_i$ when its radius $\rho$ is drawn according to $U[a, b]$:

$$\sigma^i_{a,b}(\mathbf{x}) \stackrel{\text{def}}{=} \Pr_{\rho \sim U[a,b]}(h_{i,\rho}(\mathbf{x}) = y_i) = \begin{cases} 1 & \text{if} \quad d(\mathbf{x}, \mathbf{x}_i) \leq a \\ \frac{b - d(\mathbf{x}, \mathbf{x}_i)}{b - a} & \text{if} \quad a \leq d(\mathbf{x}, \mathbf{x}_i) \leq b \\ 0 & \text{if} \quad d(\mathbf{x}, \mathbf{x}_i) \geq b_i \ . \end{cases}$$

Therefore,

$$\zeta^i_{a,b}(\mathbf{x}) \stackrel{\text{def}}{=} \Pr_{\rho \sim U[a,b]}(h_{i,\rho}(\mathbf{x}) = 1) = \begin{cases} \sigma^i_{a,b}(\mathbf{x}) & \text{if} \quad y_i = 1 \\ 1 - \sigma^i_{a,b}(\mathbf{x}) & \text{if} \quad y_i = 0 \ . \end{cases}$$

Now let $G^{\mathbf{i}}_{\mathbf{ab}}(\mathbf{x})$ denote the probability that $C_{\mathbf{i},\boldsymbol{\rho}}(\mathbf{x}) = 1$ when each $\rho_i \in \boldsymbol{\rho}$ are drawn according to $U[a_i, b_i]$. We then have

$$G^{\mathbf{i}}_{\mathbf{ab}}(\mathbf{x}) = \prod_{i \in \mathbf{i}} \zeta^i_{a_i, b_i}(\mathbf{x}) \ .$$

Consequently, the risk $R_{(\mathbf{x},y)}(G^{\mathbf{i}}_{\mathbf{ab}})$ on a single example $(\mathbf{x}, y)$ is given by $G^{\mathbf{i}}_{\mathbf{ab}}(\mathbf{x})$ if $y = 0$ and by $1 - G^{\mathbf{i}}_{\mathbf{ab}}(\mathbf{x})$ otherwise. Therefore

$$R_{(\mathbf{x},y)}(G^{\mathbf{i}}_{\mathbf{ab}}) = y(1 - G^{\mathbf{i}}_{\mathbf{ab}}(\mathbf{x})) + (1 - y)G^{\mathbf{i}}_{\mathbf{ab}}(\mathbf{x}) = (1 - 2y)(G^{\mathbf{i}}_{\mathbf{ab}}(\mathbf{x}) - y) \ .$$

Hence, the empirical risk $R_S(G^{\mathbf{i}}_{\mathbf{ab}})$ of the Gibbs classifier $G^{\mathbf{i}}_{\mathbf{ab}}$ is given by

$$R_S(G^{\mathbf{i}}_{\mathbf{ab}}) = \frac{1}{m - |\mathbf{i}|} \sum_{j \in \bar{\mathbf{i}}} (1 - 2y_j)(G^{\mathbf{i}}_{\mathbf{ab}}(\mathbf{x}_j) - y_j) \ .$$

From this expression we see that $R_S(G^{\mathbf{i}}_{\mathbf{ab}})$ is small when $G^{\mathbf{i}}_{\mathbf{ab}}(\mathbf{x}_j) \to y_j$ $\forall j \in \bar{\mathbf{i}}$. Training points where $G^{\mathbf{i}}_{\mathbf{ab}}(\mathbf{x}_j) \approx 1/2$ should therefore be avoided.

The PAC-Bayes theorem below provides a risk bound for the Gibbs classifier $G^{\mathbf{i}}_{\mathbf{ab}}$. Since the Bayes classifier $B^{\mathbf{i}}_{\mathbf{ab}}$ just performs a majority vote under the same posterior distribution as the one used by $G^{\mathbf{i}}_{\mathbf{ab}}$, we have that $B^{\mathbf{i}}_{\mathbf{ab}}(\mathbf{x}) = 1$ iff $G^{\mathbf{i}}_{\mathbf{ab}}(\mathbf{x}) > 1/2$. From the above definitions, note that the decision surface of the Bayes classifier, given by $G^{\mathbf{i}}_{\mathbf{ab}}(\mathbf{x}) = 1/2$, differs from the decision surface of classifier $C_{\mathbf{i}\boldsymbol{\rho}}$ when $\rho_i = (a_i + b_i)/2$ $\forall i \in \mathbf{i}$. In fact there does not exists any classifier $C_{\mathbf{i}\boldsymbol{\rho}}$ that has the same decision surface as Bayes classifier $B^{\mathbf{i}}_{\mathbf{ab}}$. From the relation between $B^{\mathbf{i}}_{\mathbf{ab}}$ and $G^{\mathbf{i}}_{\mathbf{ab}}$, it also follows that $R_{(\mathbf{x},y)}(B^{\mathbf{i}}_{\mathbf{ab}}) \leq 2R_{(\mathbf{x},y)}(G^{\mathbf{i}}_{\mathbf{ab}})$ for any $(\mathbf{x}, y)$. Consequently, $R(B^{\mathbf{i}}_{\mathbf{ab}}) \leq 2R(G^{\mathbf{i}}_{\mathbf{ab}})$. Hence, we have the following theorem.

**Theorem 2** *Given all our previous definitions, for any $\delta \in (0, 1]$, for any $p$ satisfying $\sum_{d=0}^{m} p(d) = 1$, and for any fixed distance value $R$, we have:*

$$\Pr_{S \sim D^m}\left( \forall \mathbf{i}, \mathbf{a}, \mathbf{b} \colon R(G^{\mathbf{i}}_{\mathbf{ab}}) \leq \sup\left\{ \epsilon \colon \mathrm{kl}(R_S(G^{\mathbf{i}}_{\mathbf{ab}}) \| \epsilon) \leq \frac{1}{m - |\mathbf{i}|}\left[ \ln\binom{m}{|\mathbf{i}|} + \right. \right. \right.$$

$$\left. \left. \left. + \ln\left(\frac{1}{p(|\mathbf{i}|)}\right) + \sum_{i \in \mathbf{i}} \ln\left(\frac{R}{b_i - a_i}\right) + \ln\frac{m+1}{\delta} \right] \right\} \right) \geq 1 - \delta \ .$$

*Furthermore:* $R(B^{\mathbf{i}}_{\mathbf{ab}}) \leq 2R(G^{\mathbf{i}}_{\mathbf{ab}})$ $\quad \forall \mathbf{i}, \mathbf{a}, \mathbf{b}.$

Recall that the KL divergence is small for small values of $|\mathbf{i}|$ (whenever $p(|\mathbf{i}|)$ is not too small) and for large margin values $(b_i - a_i)$. Furthermore, the Gibbs empirical risk $R_S(G_{\mathbf{ab}}^{\mathbf{i}})$ is small when the training points are located far away from the Bayes decision surface $G_{\mathbf{ab}}^{\mathbf{i}}(\mathbf{x}) = 1/2$ (with $G_{\mathbf{ab}}^{\mathbf{i}}(\mathbf{x}_j) \to y_j \;\forall j \in \bar{\mathbf{i}}$). *Consequently, the Gibbs classifier with the smallest guarantee of risk should perform a non trivial margin-sparsity tradeoff.*

## 4  A Soft Greedy Learning Algorithm

Theorem 2 suggests that the learner should try to find the Bayes classifier $B_{\mathbf{ab}}^{\mathbf{i}}$ that uses a small number of balls (*i.e.*, a small $|\mathbf{i}|$), each with a large separating margin $(b_i - a_i)$, while keeping the empirical Gibbs risk $R_S(G_{\mathbf{ab}}^{\mathbf{i}})$ at a low value. To achieve this goal, we have adapted the greedy algorithm for the set covering machine (SCM) proposed by Marchand and Shawe-Taylor (2002). It consists of choosing the (Boolean-valued) feature $i$ with the largest *utility* $U_i$ defined as $U_i = |N_i| - p\,|P_i|$, where $N_i$ is the set of negative examples covered (classified as 0) by feature $i$, $P_i$ is the set of positive examples misclassified by this feature, and $p$ is a learning parameter that gives a penalty $p$ for each misclassified positive example. Once the feature with the largest $U_i$ is found, we remove $N_i$ and $P_i$ from the training set $S$ and then repeat (on the remaining examples) until either no more negative examples are present or that a maximum number of features has been reached.

In our case, however, we need to keep the Gibbs risk on $S$ low instead of the risk of a deterministic classifier. Since the Gibbs risk is a "soft measure" that uses the piece-wise linear functions $\sigma_{a,b}^i$ instead of "hard" indicator functions, we need a "softer" version of the utility function $U_i$. Indeed, a negative example that falls in the linear region of a $\sigma_{a,b}^i$ is in fact partly covered. Following this observation, let $\mathbf{k}$ be the vector of indices of the examples that we have used as ball centers so far for the construction of the classifier. Let us first define the *covering value* $\mathcal{C}(G_{\mathbf{ab}}^{\mathbf{k}})$ of $G_{\mathbf{ab}}^{\mathbf{k}}$ by the "amount" of negative examples assigned to class 0 by $G_{\mathbf{ab}}^{\mathbf{k}}$:

$$\mathcal{C}(G_{\mathbf{ab}}^{\mathbf{k}}) \;\stackrel{\text{def}}{=}\; \sum_{j \in \bar{\mathbf{k}}} (1 - y_j)\left[1 - G_{\mathbf{ab}}^{\mathbf{k}}(\mathbf{x}_j)\right] \ .$$

We also define the *positive-side error* $\mathcal{E}(G_{\mathbf{ab}}^{\mathbf{k}})$ of $G_{\mathbf{ab}}^{\mathbf{k}}$ as the "amount" of positive examples assigned to class 0 :

$$\mathcal{E}(G_{\mathbf{ab}}^{\mathbf{k}}) \;\stackrel{\text{def}}{=}\; \sum_{j \in \bar{\mathbf{k}}} y_j \left[1 - G_{\mathbf{ab}}^{\mathbf{k}}(\mathbf{x}_j)\right] \ .$$

We now want to add another ball, centered on an example with index $i$, to obtain a new vector $\mathbf{k}'$ containing this new index in addition to those present in $\mathbf{k}$. Hence, we now introduce the *covering contribution* of ball $i$ (centered on $\mathbf{x}_i$) as

$$\mathcal{C}_{\mathbf{ab}}^{\mathbf{k}}(i) \;\stackrel{\text{def}}{=}\; \mathcal{C}(G_{\mathbf{a'b'}}^{\mathbf{k}'}) - \mathcal{C}(G_{\mathbf{ab}}^{\mathbf{k}})$$

$$= (1 - y_i)\left[1 - \zeta_{a_i,b_i}^i(\mathbf{x}_i)\, G_{\mathbf{ab}}^{\mathbf{k}}(\mathbf{x}_i)\right] + \sum_{j \in \bar{\mathbf{k}'}} (1 - y_j)\left[1 - \zeta_{a_i,b_i}^i(\mathbf{x}_j)\right] G_{\mathbf{ab}}^{\mathbf{k}}(\mathbf{x}_j) \ ,$$

and the *positive-side error contribution* of ball $i$ as

$$\mathcal{E}_{\mathbf{ab}}^{\mathbf{k}}(i) \;\stackrel{\text{def}}{=}\; \mathcal{E}(G_{\mathbf{a'b'}}^{\mathbf{k}'}) - \mathcal{E}(G_{\mathbf{ab}}^{\mathbf{k}})$$

$$= y_i\left[1 - \zeta_{a_i,b_i}^i(\mathbf{x}_i)\, G_{\mathbf{ab}}^{\mathbf{k}}(\mathbf{x}_i)\right] + \sum_{j \in \bar{\mathbf{k}'}} y_j\left[1 - \zeta_{a_i,b_i}^i(\mathbf{x}_j)\right] G_{\mathbf{ab}}^{\mathbf{k}}(\mathbf{x}_j) \ .$$

Typically, the covering contribution of ball $i$ should increase its "utility" and its positive-side error should decrease it. Hence, we define the *utility $U_{\mathbf{ab}}^{\mathbf{k}}(i)$ of adding ball $i$ to $G_{\mathbf{ab}}^{\mathbf{k}}$* as

$$U_{\mathbf{ab}}^{\mathbf{k}}(i) \quad \overset{\mathrm{def}}{=} \quad \mathcal{C}_{\mathbf{ab}}^{\mathbf{k}}(i) - p\mathcal{E}_{\mathbf{ab}}^{\mathbf{k}}(i) \; ,$$

where parameter $p$ represents the *penalty* of misclassifying a positive example. For a fixed value of $p$, the "soft greedy" algorithm simply consists of adding, to the current Gibbs classifier, a ball with maximum added utility until either the maximum number of possible features (balls) has been reached or that all the negative examples have been (totally) covered. It is understood that, during this soft greedy algorithm, we can remove an example $(\mathbf{x}_j, y_j)$ from $S$ whenever it is totally covered. This occurs whenever $G_{\mathbf{ab}}^{\mathbf{k}}(\mathbf{x}_j) = 0$.

The term $\sum_{i \in \mathbf{i}} \ln(R/(b_i - a_i))$, present in the risk bound of Theorem 2, favors "soft balls" having large margins $b_i - a_i$. Hence, we introduce a *margin parameter $\gamma \geq 0$* that we use as follows. At each greedy step, we first search among balls having $b_i - a_i = \gamma$. Once such a ball, of center $\mathbf{x}_i$, having maximum utility has been found, we try to increase further its utility be searching among all possible values of $a_i$ and $b_i > a_i$ while keeping its center $\mathbf{x}_i$ fixed[2]. Both $p$ and $\gamma$ will be chosen by cross validation on the training set.

We conclude this section with an analysis of the running time of this soft greedy learning algorithm for fixed $p$ and $\gamma$. For each potential ball center, we first sort the $m - 1$ other examples with respect to their distances from the center in $O(m \log m)$ time. Then, for this center $\mathbf{x}_i$, the set of $a_i$ values that we examine are those specified by the distances (from $\mathbf{x}_i$) of the $m - 1$ sorted examples[3]. Since the examples are sorted, it takes time $\in O(km)$ to compute the covering contributions and the positive-side error *for all* the $m - 1$ values of $a_i$. Here $k$ is the largest number of examples falling into the margin. We are always using small enough $\gamma$ values to have $k \in O(\log m)$ since, otherwise, the results are terrible. It therefore takes time $\in O(m \log m)$ to compute the utility values of all the $m - 1$ different balls of a given center. This gives a time $\in O(m^2 \log m)$ to compute the utilities for all the possible $m$ centers. Once a ball with a largest utility value has been chosen, we then try to increase further its utility by searching among $O(m^2)$ pair values for $(a_i, b_i)$. We then remove the examples covered by this ball and repeat the algorithm on the remaining examples. It is well known that greedy algorithms of this kind have the following guarantee: if there exist $r$ balls that covers all the $m$ examples, the greedy algorithm will find at most $r \ln(m)$ balls. Since we almost always have $r \in O(1)$, the running time of the whole algorithm will almost always be $\in O(m^2 \log^2(m))$.

## 5   Empirical Results on Natural Data

We have compared the new PAC-Bayes learning algorithm (called here SCM-PB), with the old algorithm (called here SCM). Both of these algorithms were also compared with the SVM equipped with a RBF kernel of variance $\sigma^2$ and a soft margin parameter $C$. Each SCM algorithm used the $L_2$ metric since this is the metric present in the argument of the RBF kernel. However, in contrast with Laviolette et al. (2005), each SCM was constrained to use only balls having centers of the same class (negative for conjunctions and positive for disjunctions).

Table 1: SVM and SCM results on UCI data sets.

| Data Set | | | SVM results | | | | SCM | | SCM-PB | | |
|---|---|---|---|---|---|---|---|---|---|---|---|
| Name | train | test | $C$ | $\sigma^2$ | SVs | errs | b | errs | b | $\gamma$ | errs |
| breastw | 343 | 340 | 1 | 5 | 38 | 15 | 1 | 12 | 4 | .08 | 10 |
| bupa | 170 | 175 | 2 | .17 | 169 | 66 | 5 | 62 | 6 | .1 | 67 |
| credit | 353 | 300 | 100 | 2 | 282 | 51 | 3 | 58 | 11 | .09 | 55 |
| glass | 107 | 107 | 10 | .17 | 51 | 29 | 5 | 22 | 16 | .04 | 19 |
| heart | 150 | 147 | 1 | .17 | 64 | 26 | 1 | 23 | 1 | 0 | 28 |
| haberman | 144 | 150 | 2 | 1 | 81 | 39 | 1 | 39 | 1 | .2 | 38 |
| USvotes | 235 | 200 | 1 | 25 | 53 | 13 | 10 | 27 | 18 | .14 | 12 |

Each algorithm was tested the UCI data sets of Table 1. Each data set was randomly split in two parts. About half of the examples was used for training and the remaining set of examples was used for testing. The corresponding values for these numbers of examples are given in the "train" and "test" columns of Table 1. The learning parameters of all algorithms were determined from the training set *only*. The parameters $C$ and $\gamma$ for the SVM were determined by the 5-fold cross validation (CV) method performed on the training set. The parameters that gave the smallest 5-fold CV error were then used to train the SVM on the whole training set and the resulting classifier was then run on the testing set. Exactly the same method (with the same 5-fold split) was used to determine the learning parameters of both SCM and SCM-PB.

The SVM results are reported in Table 1 where the "SVs" column refers to the number of support vectors present in the final classifier and the "errs" column refers to the number of classification errors obtained on the testing set. This notation is used also for all the SCM results reported in Table 1. In addition to this, the "b" and "$\gamma$" columns refer, respectively, to the number of balls and the margin parameter (divided by the average distance between the positive and the negative examples). The results reported for SCM-PB refer to the Bayes classifier only. The results for the Gibbs classifier are similar. We observe that, except for bupa and heart, the generalization error of SCM-PB was always smaller than SCM. However, the only significant difference occurs on USvotes. We also observe that SCM-PB generally sacrifices sparsity (compared to SCM) to obtain some margin $\gamma > 0$.

## Footnotes

[1] We assume that the examples in $S_{\mathbf{i}}$ are ordered as in $S$ so that the $k$th radius value in $\boldsymbol{\rho}$ is assigned to the $k$th example in $S_{\mathbf{i}}$.

[2]The possible values for $a_i$ and $b_i$ are defined by the location of the training points.

[3]Recall that for each value of $a_i$, the value of $b_i$ is set to $a_i + \gamma$ at this stage.

# References

B. E. Boser, I. M. Guyon, and V. N. Vapnik. A training algorithm for optimal margin classifiers. In *Proceedings of the 5th Annual ACM Workshop on Computational Learning Theory*, pages 144–152. ACM Press, 1992.

John Langford. Tutorial on practical prediction theory for classification. *Journal of Machine Learning Research*, 6:273–306, 2005.

François Laviolette and Mario Marchand. PAC-Bayes risk bounds for sample-compressed Gibbs classifiers. *Proceedings of the 22nth International Conference on Machine Learning (ICML 2005)*, pages 481–488, 2005.

François Laviolette, Mario Marchand, and Mohak Shah. Margin-sparsity trade-off for the set covering machine. *Proceedings of the $16^{th}$ European Conference on Machine Learning (ECML 2005); Lecture Notes in Artificial Intelligence*, 3720:206–217, 2005.

Mario Marchand and John Shawe-Taylor. The set covering machine. *Journal of Machine Learning Reasearch*, 3:723–746, 2002.

David McAllester. Some PAC-Bayesian theorems. *Machine Learning*, 37:355–363, 1999a.

David A. McAllester. Pac-bayesian model averaging. In *COLT*, pages 164–170, 1999b.
